# Multilayer neural networks: one or two hidden layers?

**G. Brightwell**
Dept of Mathematics
LSE, Houghton Street
London WC2A 2AE, U.K.

**C. Kenyon, H. Paugam-Moisy**
LIP, URA 1398 CNRS
ENS Lyon, 46 allée d'Italie
F69364 Lyon cedex, FRANCE

## Abstract

We study the number of hidden layers required by a multilayer neural network with threshold units to compute a function $f$ from $\mathcal{R}^d$ to $\{0, 1\}$. In dimension $d = 2$, Gibson characterized the functions computable with just one hidden layer, under the assumption that there is no "multiple intersection point" and that $f$ is only defined on a compact set. We consider the restriction of $f$ to the neighborhood of a multiple intersection point or of infinity, and give necessary and sufficient conditions for it to be locally computable with one hidden layer. We show that adding these conditions to Gibson's assumptions is not sufficient to ensure global computability with one hidden layer, by exhibiting a new non-local configuration, the "critical cycle", which implies that $f$ is not computable with one hidden layer.

## 1 INTRODUCTION

The number of hidden layers is a crucial parameter for the architecture of multilayer neural networks. Early research, in the 60's, addressed the problem of exactly realizing Boolean functions with binary networks or binary multilayer networks. On the one hand, more recent work focused on approximately realizing real functions with multilayer neural networks with one hidden layer [6, 7, 11] or with two hidden units [2]. On the other hand, some authors [1, 12] were interested in finding bounds on the architecture of multilayer networks for exact realization of a finite set of points. Another approach is to search the minimal architecture of multilayer networks for exactly realizing real functions, from $\mathcal{R}^d$ to $\{0, 1\}$. Our work, of the latter kind, is a continuation of the effort of [4, 5, 8, 9] towards characterizing the real dichotomies which can be exactly realized with a single hidden layer neural network composed of threshold units.

## 1.1 NOTATIONS AND BACKGROUND

A finite set of hyperplanes $\{H_i\}_{1 \leq i \leq h}$ defines a partition of the $d$-dimensional space into convex polyhedral open regions, the union of the $H_i$'s being neglected as a subset of measure zero. A *polyhedral dichotomy* is a function $f : \mathcal{R}^d \rightarrow \{0, 1\}$, obtained by associating a class, equal to 0 or to 1, to each of those regions. Thus both $f^{-1}(0)$ and $f^{-1}(1)$ are unions of a finite number of convex polyhedral open regions. The $h$ hyperplanes which define the regions are called the *essential hyperplanes* of $f$. A point $P$ is an *essential point* if it is the intersection of some set of essential hyperplanes.

In this paper, all *multilayer networks* are supposed to be feedforward neural networks of threshold units, fully interconnected from one layer to the next, without skipping interconnections. A network is said to *realize* a function $f : \mathcal{R}^d \rightarrow \{0, 1\}$ if, for an input vector $x$, the network output is equal to $f(x)$, almost everywhere in $\mathcal{R}^d$. The functions realized by our multilayer networks are the polyhedral dichotomies.

By definition of threshold units, each unit of the first hidden layer computes a binary function $y_j$ of the real inputs $(x_1, \ldots, x_d)$. Therefore, subsequent layers compute a Boolean function. Since any Boolean function can be written in DNF-form, two hidden layers are sufficient for a multilayer network to realize any polyhedral dichotomy. Two hidden layers are sometimes also necessary, e.g. for realizing the "four-quadrant" dichotomy which generalizes the XOR function [4].

For all $j$, the $j^{th}$ unit of the first hidden layer can be seen as separating the space by the hyperplane $H_j : \sum_{i=1}^{d} w_{ij} x_i = \theta_j$. Hence the first hidden layer necessarily contains at least one hidden unit for each essential hyperplane of $f$. Thus each region $R$ can be labelled by a binary number $y = (y_1, \ldots, y_h)$ (see [5]). The $j^{th}$ digit $y_j$ will be denoted by $H_j(R)$.

Usually there are fewer than $2^h$ regions and not all possible labels actually exist. The *Boolean family* $\mathcal{B}_f$ of a polyhedral dichotomy $f$ is defined to be the set of all Boolean functions on $h$ variables which are equal to $f$ on all the existing labels.

## 1.2 PREVIOUS RESULTS

It is straightforward that all polyhedral dichotomies which have at least one linearly separable function in their Boolean family can be realized by a one-hidden-layer network. However the converse is far from true. A counter-example was produced in [5]: adding extra hyperplanes (i.e. extra units on the first hidden layer) can eliminate the need for a second hidden layer. Hence the problem of finding a minimal architecture for realizing dichotomies cannot be reduced to the neural computation of Boolean functions. Finding a generic description of all the polyhedral dichotomies which can be realized exactly by a one-hidden-layer network is still an open problem. This paper is a new step towards its resolution.

One approach consists of finding geometric configurations which imply that a function is not realizable with a single hidden layer. There are three known such geometric configurations: the XOR-situation, the XOR-bow-tie and the XOR-at-infinity (see Figure 1).

A polyhedral dichotomy is said to be *in an XOR-situation* iff one of its essential hyperplanes $H$ is *inconsistent*, i.e. if there are four regions $B, B', W, W'$ such that $B$ and $B'$ are in class 1, $W$ and $W'$ are in class 0, $B$ and $W'$ are on one side of $H$, $B'$ and $W$ are on the other side of $H$, and $B$ and $W$ are adjacent along $H$, as well as $B'$ and $W'$.

Given a point $P$, two regions containing $P$ in their closure are called *opposite with respect to $P$* if they are in different halfspaces w.r.t. all essential hyperplanes going through $P$. A polyhedral dichotomy is said to be *in an XOR-bow-tie* iff there exist four distinct regions $B, B', W, W'$, such that $B$ and $B'$, both in class 1 (resp. $W$ and $W'$, both in class 0), are opposite with respect to point $P$.

The third configuration is the *XOR-at-infinity*, which is analogous to the XOR-bow-tie at a point $\infty$ added to $\mathcal{R}^d$. There exist four distinct unbounded regions $B, B'$ (in class 1), $W, W'$ (in class 0) such that, for every essential hyperplane $H$, either all of them are on the same side of $H$ (e.g. the horizontal line), or $B$ and $B'$ are on opposite sides of $H$, and $W$ and $W'$ are on opposite sides of $H$ (see [3]).

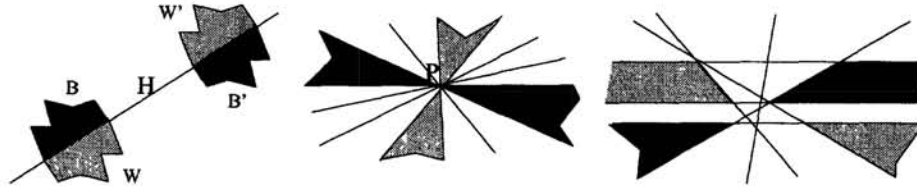

Figure 1: Geometrical representation of XOR-situation, XOR-bow-tie and XOR-at-infinity in the plane (black regions are in class 1, grey regions are in class 0).

**Theorem 1** *If a polyhedral dichotomy $f$, from $\mathcal{R}^d$ to $\{0, 1\}$, can be realized by a one-hidden-layer network, then it cannot be in an XOR-situation, nor in an XOR-bow-tie, nor in an XOR-at-infinity.*

The proof can be found in [5] for the XOR-situation, in [13] for the XOR-bow-tie, and in [5] for the XOR-at-infinity.

Another research direction, implying a function is realizable by a single hidden layer network, is based on the universal approximator property of one-hidden-layer networks, applied to intermediate functions obtained constructively adding extra hyperplanes to the essential hyperplanes of $f$. This direction was explored by Gibson [9], but there are virtually no results known beyond two dimensions. Gibson's result can be reformulated as follows:

**Theorem 2** *If a polyhedral dichotomy $f$ is defined on a compact subset of $\mathcal{R}^2$, if $f$ is not in an XOR-situation, and if no three essential hyperplanes (lines) intersect, then $f$ is realizable with a single hidden layer network.*

Unfortunately Gibson's proof is not constructive, and extending it to remove some of the assumptions or to go to higher dimensions seems challenging. Both XOR-bow-tie and XOR-at-infinity are excluded by his assumptions of compactness and no multiple intersections. In the next section, we explore the two cases which are excluded by Gibson's assumptions. We prove that, in $\mathcal{R}^2$, the XOR-bow-tie and the XOR-at-infinity are the only restrictions to local realizability.

## 2   LOCAL REALIZATION IN $\mathcal{R}^2$

### 2.1   MULTIPLE INTERSECTION

**Theorem 3** *Let $f$ be a polyhedral dichotomy on $\mathcal{R}^2$ and let $P$ be a point of multiple intersection. Let $C_P$ be a neighborhood of $P$ which does not intersect any essential hyperplane other than those going through $P$. The restriction of $f$ to $C_P$ is realizable by a one-hidden-layer network iff $f$ is not in an XOR-bow-tie at $P$.*

The proof is in three steps: first, we reorder the hyperplanes in the neighborhood of $P$, so as to get a nice looking system of inequalities; second, we apply Farkas' lemma; third, we show how an XOR-bow-tie can be deduced.

**Proof:** Let $P$ be the intersection of $k \geq 3$ essential hyperplanes of $f$. All the hyperplanes which intersect at $P$ can be renumbered and re-oriented so that the intersecting hyperplanes are totally ordered. Thus the label of the regions which have the point $P$ in their closure is very regular. If one drops all the digits corresponding to the essential hyperplanes of $f$ which do not contain $P$, the remaining part of the region labels are exactly like those of Figure 2.

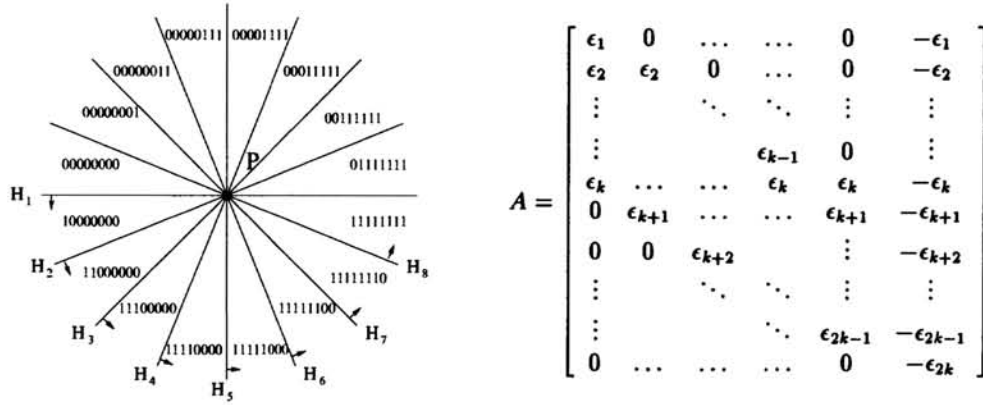

Figure 2: Labels of the regions in the neighborhood of $P$, and matrix $A$.

The problem of finding a one-hidden-layer network which realizes $f$ can be rewritten as a system of inequalities. The unknown variables are the weights $w_i$ and threshold $\theta$ of the output unit. Let $(\mathcal{S})$ denote the subsystem of inequalities obtained from the $2k$ regions which have the point $P$ in their closure. The regular numbering of these $2k$ regions allows us to write the system as follows

$$(\mathcal{S}) \begin{cases} 1 \leq i \leq k & \begin{bmatrix} \sum_{m=1}^{i} w_m & < & \theta & \text{if class 0} \\ \sum_{m=1}^{i} w_m & > & \theta & \text{if class 1} \end{bmatrix} \\ k+1 \leq i \leq 2k & \begin{bmatrix} \sum_{m=i-k+1}^{k} w_m & < & \theta & \text{if class 0} \\ \sum_{m=i-k+1}^{k} w_m & > & \theta & \text{if class 1} \end{bmatrix} \end{cases}$$

The system $(\mathcal{S})$ can be rewritten in the matrix form $Ax \leq b$, where

$$x^\mathsf{T} = [w_1, w_2, \ldots, w_k, \theta] \text{ and } b^\mathsf{T} = [b_1, b_2, \ldots, b_k, b_{k+1}, \ldots, b_{2k}]$$

where $b_i = -\epsilon$, for all $i$, and $\epsilon$ is an arbitrary small positive number. Matrix A can be seen in figure 2, where $\epsilon_j = +1$ or $-1$ depending on whether region $j$ is in class 0 or 1. The next step is to apply Farkas lemma, or an equivalent version [10], which gives a necessary and sufficient condition for finding a solution of $Ax \leq b$.

**Lemma 1 (Farkas lemma)** *There exists a vector $x \in \mathcal{R}^n$ such that $Ax \leq b$ iff there does not exist a vector $y \in \mathcal{R}^m$ such that $y^\mathsf{T} A = 0$, $y \geq 0$ and $y^\mathsf{T} b < 0$.*

Assume that $Ax \leq b$ is not solvable. Then, by Lemma 1 for $n = k+1$ and $m = 2k$, a vector $y$ can be found such that $y \geq 0$. Since in addition $y^\mathsf{T} b = -\epsilon \sum_{j=1}^{2k} y_j$, the condition $y^\mathsf{T} b < 0$ implies $(\exists j_1)\ y_{j_1} > 0$. But $y^\mathsf{T} A = 0$ is equivalent to the system

$(\mathcal{E})$ of $k + 1$ equations

$$(\mathcal{E}) \begin{cases} 1 \le i \le k & \sum_{m=i}^{i+k-1} y_{m/class\ 0} = \sum_{m=i}^{i+k-1} y_{m/class\ 1} \\ i = k+1 & \sum_{m=1}^{2k} y_{m/class\ 0} = \sum_{m=1}^{2k} y_{m/class\ 1} \end{cases}$$

Since $(\exists j_1)\ y_{j_1} > 0$, the last equation $(E_{k+1})$ of system $(\mathcal{E})$ implies that $(\exists j_2\ /\ \text{class(region } j_1) \ne \text{class(region } j_2))\ y_{j_2} > 0$. Without loss of generality, assume that $j_1$ and $j_2$ are less than $k$ and that region $j_1$ is in class 0 and region $j_2$ is in class 1. Comparing two successive equations of $(\mathcal{E})$, for $i < k$, we can write

$$(\forall \lambda \in \{0,1\})\ \sum_{(E_{i+1})} y_{m/class\ \lambda} = \sum_{(E_i)} y_{m/class\ \lambda} - y_{i/class\ \lambda} + y_{i+k/class\ \lambda}$$

Since $y_{j_1} > 0$ and region $j_1$ is in class 0, the transition from $E_{j_1}$ to $E_{j_1+1}$ implies that $y_{j_1+k} = y_{j_1} > 0$ and region $j_1 + k$, which is opposite to region $j_1$, is also in class 0. Similarly, the transition from $E_{j_2}$ to $E_{j_2+1}$ implies that both opposite regions $j_2 + k$ and $j_2$ are in class 1. These conditions are necessary for the system $(\mathcal{E})$ to have a non-negative solution and they correspond exactly to the definition of an XOR-bow-tie at point $P$. The converse comes from theorem 1.  ∎

## 2.2  UNBOUNDED REGIONS

If no two essential hyperplanes are parallel, the case of unbounded regions is exactly the same as a multiple intersection. All the unbounded regions can be labelled as on figure 2. The same argument holds for proving that, if the local system $(\mathcal{S})$ $Ax \le b$ is not solvable, then there exists an XOR-at-infinity. The case of parallel hyperplanes is more intricate because matrix $A$ is more complex. The proof requires a heavy case-by-case analysis and cannot be given in full in this paper (see [3]).

**Theorem 4** *Let $f$ be a polyhedral dichotomy on $\mathcal{R}^2$. Let $C_\infty$ be the complementary region of the convex hull of the essential points of $f$. The restriction of $f$ to $C_\infty$ is realizable by a one-hidden-layer network iff $f$ is not in an XOR-at-infinity.*

From theorems 3 and 4 we can deduce that a polyhedral dichotomy is locally realizable in $\mathcal{R}^2$ by a one-hidden-layer network iff $f$ has no XOR-bow-tie and no XOR-at-infinity. Unfortunately this result cannot be extended to the global realization of $f$ in $\mathcal{R}^2$ because more intricate distant configurations can involve contradictions in the complete system of inequalities. The object of the next section is to point out such a situation by producing a new geometric configuration, called a *critical cycle*, which implies that $f$ cannot be realized with one hidden layer.

## 3  CRITICAL CYCLES

In contrast to section 2, the results of this section hold for any dimension $d \ge 2$.

We first need some definitions. Consider a pair of regions $\{T, T'\}$ in the same class and which both contain an essential point $P$ in their closure. This pair is called *critical* with respect to $P$ and $H$ if there is an essential hyperplane $H$ going through $P$ such that $T'$ is adjacent along $H$ to the region opposite to $T$. Note that $T$ and $T'$ are both on the same side of $H$.

We define a graph $G$ whose nodes correspond to the critical pairs of regions of $f$. There is a red edge between $\{T, T'\}$ and $\{U, U'\}$ if the pairs, in different classes, are both critical with respect to the same point (e.g., $\{B_P, B'_P\}$ and $\{W_P, W'_P\}$ in figure 3). There is a green edge between $\{T, T'\}$ and $\{U, U'\}$ if the pairs are both critical with respect to the same hyperplane $H$, and either the two pairs are on the

same side of $H$, but in different classes (e.g., $\{W_P, W_P'\}$ and $\{B_Q, B_Q'\}$), or they are on different sides of $H$, but in the same class (e.g., $\{B_P, B_P'\}$ and $\{B_R, B_R'\}$).

**Definition 1** *A* critical cycle *is a cycle in graph $G$, with alternating colors.*

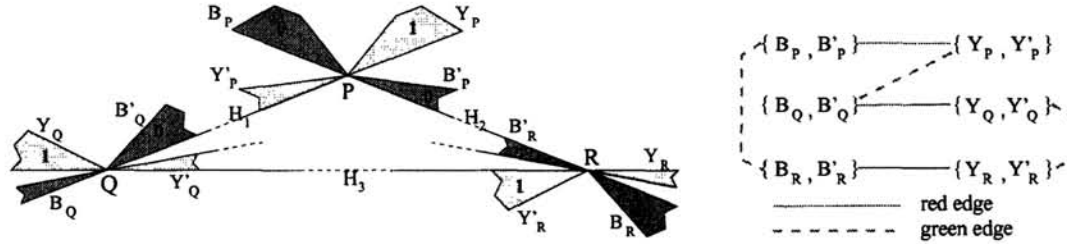

Figure 3: Geometrical configuration and graph of a critical cycle, in the plane. Note that one can augment the figure in such a way that there is no XOR-situation, no XOR-bow-tie, and no XOR-at-infinity.

**Theorem 5** *If a polyhedral dichotomy $f$, from $\mathcal{R}^d$ to $\{0, 1\}$, can be realized by a one-hidden-layer network, then it cannot have a critical cycle.*

**Proof:** For the sake of simplicity, we will restrict ourselves to doing the proof for a case similar to the example figure 3, with notation as given in that figure, but without any restriction on the dimension $d$ of $f$. Assume, for a contradiction, that $f$ has a critical cycle and can be realized by a one-hidden-layer network. Consider the sets of regions $\{B_P, B_P', B_Q, B_Q', B_R, B_R'\}$ and $\{W_P, W_P', W_Q, W_Q', W_R, W_R'\}$. Consider the regions defined by all the hyperplanes associated to the hidden layer units (in general, these hyperplanes are a large superset of the essential hyperplanes). There is a region $b_P \subseteq B_P$, whose border contains $P$ and a $(d-1)$-dimensional subset of $H_1$. Similarly we can define $b_P', \ldots, b_R', w_P, \ldots, w_R'$. Let $\mathcal{B}$ be the set of such regions which are in class 1 and $\mathcal{W}$ be the set of such regions in class 0.

Let $H$ be the hyperplane associated to one of the hidden units. For $T$ a region, let $H(T)$ be the digit label of $T$ w.r.t. $H$, i.e. $H(T) = 1$ or 0 according to whether $T$ is above or below $H$ (cf. section 1.1). We do a case-by-case analysis.

If $H$ does not go through $P$, then $H(b_P) = H(b_P') = H(w_P) = H(w_P')$; similar equalities hold for lines not going through $Q$ or $R$. If $H$ goes through $P$ but is not equal to $H_1$ or to $H_2$, then, from the viewpoint of $H$, things are as if $b_P'$ was opposite to $b_P$, and $w_P'$ was opposite to $w_P$, so the two regions of each pair are on different sides of $H$, and so $H(b_P) + H(b_P') = H(w_P) + H(w_P') = 1$; similar equalities hold for hyperplanes going through $Q$ or $R$. If $H = H_1$, then we use the fact that there is a green edge between $\{W_P, W_P'\}$ and $\{B_Q, B_Q'\}$, meaning in the case of the figure that all four regions are on the same side of $H_1$ but in different classes. Then $H(b_P) + H(b_P') + H(b_Q) + H(b_Q') = H(w_P) + H(w_P') + H(w_Q) + H(w_Q')$. In fact, this equality would also hold in the other case, as can easily be checked. Thus for all $H$, we have $\sum_{b \in \mathcal{B}} H(b) = \sum_{w \in \mathcal{W}} H(w)$. But such an equality is impossible: since each $b$ is in class 1 and each $w$ is in class 0, this implies a contradiction in the system of inequalities and $f$ cannot be realized by a one-hidden-layer network.

Obviously there can exist cycles of length longer than 3, but the extension of the proof is straightforward. ∎

# 4  CONCLUSION AND PERSPECTIVES

This paper makes partial progress towards characterizing functions which can be realized by a one-hidden-layer network, with a particular focus on dimension 2. Higher dimensions are more challenging, and it is difficult to even propose a conjecture: new cases of inconsistency emerge in subspaces of intermediate dimension. Gibson gives an example of an inconsistent line (dimension 1) resulting of its intersection with two hyperplanes (dimension 2) which are not inconsistent in $\mathcal{R}^3$.

The principle of using Farkas lemma for proving local realizability still holds but the matrix $A$ becomes more and more complex. In $\mathcal{R}^d$, even for $d = 3$, the labelling of the regions, for instance around a point $P$ of multiple intersection, can become very complex.

In conclusion, it seems that neither the topological method of Gibson, nor our algebraic point of view, can easily be extended to higher dimensions. Nevertheless, we conjecture that in dimension 2, a function can be realized by a one-hidden-layer network iff it does not have any of the four forbidden types of configurations: XOR-situation, XOR-bow-tie, XOR-at-infinity, and critical cycle.

### Acknowledgements

This work was supported by European Esprit III Project $n^0$ 8556, NeuroCOLT.

# References

[1] E. B. Baum. On the capabilities of multilayer perceptrons. *Journal of Complexity*, 4:193–215, 1988.

[2] E. K. Blum and L. K. Li. Approximation theory and feedforward networks. *Neural Networks*, 4(4):511–516, 1991.

[3] G. Brightwell, C. Kenyon, and H. Paugam-Moisy. Multilayer neural networks: one or two hidden layers? Research Report 96-37, LIP, ENS Lyon, 1996.

[4] M. Cosnard, P. Koiran, and H. Paugam-Moisy. Complexity issues in neural network computations. In I. Simon, editor, *Proc. of LATIN'92*, volume 583 of *LNCS*, pages 530–544. Springer Verlag, 1992.

[5] M. Cosnard, P. Koiran, and H. Paugam-Moisy. A step towards the frontier between one-hidden-layer and two-hidden layer neural networks. In I. Simon, editor, *Proc. of IJCNN'93-Nagoya*, volume 3, pages 2292–2295. Springer Verlag, 1993.

[6] G. Cybenko. Approximation by superpositions of a sigmoidal function. *Math. Control, Signal Systems*, 2:303–314, October 1988.

[7] K. Funahashi. On the approximate realization of continuous mappings by neural networks. *Neural Networks*, 2(3):183–192, 1989.

[8] G. J. Gibson. A combinatorial approach to understanding perceptron decision regions. *IEEE Trans. Neural Networks*, 4:989–992, 1993.

[9] G. J. Gibson. Exact classification with two-layer neural nets. *Journal of Computer and System Science*, 52(2):349–356, 1996.

[10] M. Grötschel, L. Lovsz, and A. Schrijver. *Geometric Algorithms and Combinatorial Optimization*. Springer-Verlag, Berlin, Heidelberg, 1988.

[11] K. Hornik, M. Stinchcombe, and H. White. Multilayer feedforward networks are universal approximators. *Neural Networks*, 2(5):359–366, 1989.

[12] S.-C. Huang and Y.-F. Huang. Bounds on the number of hidden neurones in multilayer perceptrons. *IEEE Trans. Neural Networks*, 2:47–55, 1991.

[13] P. J. Zweitering. *The complexity of multi-layered perceptrons*. PhD thesis, Technische Universiteit Eindhoven, 1994.
